# Learning Complete Protein Representation by Dynamically Coupling of Sequence and Structure

**Bozhen Hu**[1,2*]**, Cheng Tan**[2*]**, Jun Xia**[2]**, Yue Liu**[3]**, Lirong Wu**[2]**,**
**Jiangbin Zheng**[2]**, Yongjie Xu**[2]**, Yufei Huang**[2]**, Stan Z. Li**[2†]

[1]Zhejiang University    [2]Westlake University    [3]National University of Singapore
{hubozhen, tancheng, stan.zq.li}@westlake.edu.cn

## Abstract

Learning effective representations is imperative for comprehending proteins and deciphering their biological functions. Recent strides in language models and graph neural networks have empowered protein models to harness primary or tertiary structure information for representation learning. Nevertheless, the absence of practical methodologies to appropriately model intricate inter-dependencies between protein sequences and structures has resulted in embeddings that exhibit low performance on tasks such as protein function prediction. In this study, we introduce CoupleNet, a novel framework designed to interlink protein sequences and structures to derive informative protein representations. CoupleNet integrates multiple levels and scales of features in proteins, encompassing residue identities and positions for sequences, as well as geometric representations for tertiary structures from both local and global perspectives. A two-type dynamic graph is constructed to capture adjacent and distant sequential features and structural geometries, achieving completeness at the amino acid and backbone levels. Additionally, convolutions are executed on nodes and edges simultaneously to generate comprehensive protein embeddings. Experimental results on benchmark datasets showcase that CoupleNet outperforms state-of-the-art methods, exhibiting particularly superior performance in low-sequence similarities scenarios, adeptly identifying infrequently encountered functions and effectively capturing remote homology relationships in proteins.

## 1  Introduction

Proteins, the fundamental building blocks of life, serve crucial roles across a diverse array of applications, ranging from therapeutics to materials. Comprising 20 distinct amino acids linked by peptide bonds, proteins form intricate sequences that dictate their one-dimensional (1D) structure, ultimately determining their biochemical functions [1]. Due to recent progress in protein sequencing [2], massive numbers of protein sequences are now available. For example, the UniProt database, housing over 200 million protein sequences, has become a valuable resource for research [3]. Notably, the development of large-scale language models (LMs) in natural language processing has substantially benefited protein research owing to similarities between human languages and protein sequences [4–6]. For instance, models like ProtTrans [7] and ESM-series [8, 9] have proven the successful utility of protein LMs to process protein sequences.

Thanks to the recent significant progress made by AlphaFold2 [10] in three-dimensional (3D) structure prediction, a large number of protein structures from their sequence data are now made available. The latest release of AlphaFold protein structure database [11] provides broad coverage of UniProt [3]. Recently proposed structure-based protein encoders become to utilize geometric features [12–14],

e.g., ProNet [15] learns representations of proteins with 3D structures at different levels, like the amino acid, backbone or all-atom levels. Concurrently, methods employing graph neural networks (GNNs) and LMs (LSTMs or attention models) [14, 16, 17], such as GearNet [14], have been developed to process both sequence and structure information.

The 1D sequence and 3D structure of a protein provide different types of information, the discrete sequential orders, residue types, and coordinates, as shown in Figure 5 and Figure 6 in Appendix A. Models can learn coevolutionary and geometric information from sequences and structures, for example, whether residues contact or not. Although a protein's sequence determines its structure, various works have demonstrated the effectiveness of learning from either sequences or structures [9, 12, 18]. However, directly fusing representations from sequence encoder and structure encoder cannot explore their relationships, and current protein GNN methods have drawbacks in integrating such sequential and structural information. In detail, the information propagation is difficult for long-range dependencies in large protein graphs, and messages attenuate over many rounds of passing in GNNs, although there are several works aiming to tackle such a problem [19, 20]. Besides, message passing typically assumes localized neighborhood relationships, but amino acid interactions can be complex and long-range. We need to consider the structural and chemical properties of a residue that are highly dependent on surrounding neighbors, and capturing different conformers requires modeling the entire protein structure holistically, as the conformation of an amino acid is constrained by steric and hydrogen bonding with nearby residues, and the conformers correspond to the same protein, adopting slightly different 3D structures. Therefore, a proper protein sequence-structure modeling method is necessary and important to recognize these challenges and factors to obtain comprehensive and effective representations.

In this work, we model the relative positions of residues in the sequence and the spatial arrangement of atoms in Euclidean space simultaneously. We propose CoupleNet to construct two categories of graphs dynamically to cover the multiple scales of sequential features and structural geometries, which achieve completeness at the residue and backbone levels. Such global completeness is theoretically guaranteed to incorporate 3D information completely without information loss, while the local view would miss the long-range effects of subtle conformational changes happening distantly. For instance, the open and closed conformers of an enzyme may have similar local binding pockets but differ in global clamshell arrangement [21]. In order to better capture the local and global relationships and relieve the problems that exist in deep GNNs, we dynamically build new protein graphs in different conditions based on the depth of the network. For feature fusing, we take advantage of graph convolutions, performing node and edge convolutions simultaneously rather than passing messages separately on nodes and edges. Thus, the contributions of this paper are threefold: (1) A novel two-graph-based approach for modeling sequential and 3D geometric features, ensuring global completeness in protein representation. (2) The proposal of CoupleNet, which performs concurrent convolutions on nodes and edges, effectively integrating protein sequence and structure. The dynamically changed graphs can better model the node-edge relationships and utilize the intrinsic associations between sequences and structures. (3) Empirical validation showcases the superior performance of the proposed model compared to current mainstream protein representation learning methods across diverse tasks, including protein fold classification, enzyme reaction classification, Gene Ontology (GO) term prediction, and Enzyme Commission (EC) number prediction. Our experiments reveal that this method excels in predicting functions that rarely appear, effectively captures protein remote homology relationships.

## 2   Related Work

**Protein Representation Learning.**   Protein representation learning has emerged as a dynamic and promising field within biology, playing a crucial role in diverse downstream applications in protein science. Given the multifaceted nature of protein structures, current methodologies predominantly fall into three categories: protein LMs tailored for sequences, structure models emphasizing geometry, and hybrid approaches seamlessly integrating both aspects. Considering proteins as sequences of amino acids, akin to the structure of human languages, TAPE [22] establishes a benchmark for a variety of protein models, including 1D CNN, LSTM, and Transformer architectures. Elnaggar et al. have successfully trained six transformer variants, such as ProtBert and ProtT5, on extensive amino acid sequences. Similarly, the ESM-series [8, 9, 23] adopt a transformer architecture and a masked language modeling strategy, achieving robust representations through training on large-scale

databases. Besides, several methods aim to encode the spatial information of protein structures using techniques such as convolutional neural networks (CNNs) [24], or GNNs [17, 25, 26]. For instance, SPROF [27] employs distance maps to predict protein sequence profiles, while IEConv [12] introduces a convolution operator to capture relevant structural levels. GVP-GNN [26] designs the geometric vector perceptrons (GVP) to learn both scalar and vector features in an equivariant and invariant manner. ProNet [15] learns hierarchical protein representations at multiple tertiary structure levels of granularity. Additionally, CDConv [28] introduces continuous-discrete convolution, utilizing irregular and regular approaches to model both geometry and sequence structures. A protein clustering method [29] is proposed by applying an iterative clustering strategy to group the nodes into clusters based on their 1D and 3D positions and assigned scores to obtain hierarchical protein representations. Moreover, some protein learning methods concurrently model multiple levels of structures [14, 28, 30], and PromtProtein [31] adopts a prompt-guided multi-task learning strategy for incorporating various protein structures.

**Complete Message Passing.**    While SphereNet [32] introduces a spherical message passing scheme for precise 3D molecular learning, ensuring completeness within the edge-based 1-hop local neighborhood, this completeness does not extend to the entire 3D graph. Building upon this limitation, ComENet [33] innovatively incorporates rotation angles and spherical coordinates to achieve global completeness in 3D information on molecular graphs. By integrating these meticulously designed geometric representations into the established message passing scheme [34], the complete representation for a whole 3D graph is ultimately achieved [15].

Unlike these methods, we couple sequence and structure via dynamically changed graphs and different geometric representations to attain complete representations throughout the entire protein 3D graph.

# 3 Methodology

## 3.1 Preliminaries

**Notations.**    A 3D graph is represented as $G = (\mathcal{V}, \mathcal{E}, \mathcal{P})$, where $\mathcal{V} = \{v_i\}_{i=1,...,n}$ and $\mathcal{E} = \{\varepsilon_{ij}\}_{i,j=1,...,n}$ denote the vertex and edge sets with $n$ nodes, respectively, and $\mathcal{P} = \{P_i\}_{i=1,...,n}$ is the set of position matrices, where $P_i \in \mathbb{R}^{k_i \times 3}$ represents the position matrix for node $v_i$. We treat each amino acid as a graph node for a protein, then $k_i$ depends on the number of atoms in the $i$-th amino acid. The node feature matrix is $X = [\boldsymbol{x}_i]_{i=1,...,n}$, where $\boldsymbol{x}_i \in \mathbb{R}^{d_v}$ is the feature vector of node $v_i$. The edge feature matrix is $E = [\boldsymbol{e}_{ij}]_{i,j=1,...,n}$, where $\boldsymbol{e}_{ij} \in \mathbb{R}^{d_\varepsilon}$ is the feature vector of edge $\varepsilon_{ij}$. $d_v$ is the dimension of feature vector $\boldsymbol{x}_i$, and $d_\varepsilon$ denotes the dimension of $\boldsymbol{e}_{ij}$.

**Invariance and Equivariance.**    We consider affine transformations that preserve the distance between any two points, i.e., the isometric group SE(3) (refer to Appendix B) in the Euclidean space. This is called the symmetry group, and it turns out that SE(3) is the special Euclidean group that includes 3D translations and the 3D rotation group SO(3) [35, 36].

Given the function $f : \mathbb{R}^m \to \mathbb{R}^{m'}$, assuming the given symmetry group $G$ acts on $\mathbb{R}^m$ and $\mathbb{R}^{m'}$, then $f$ is G-equivariant if,

$$f(T_g\boldsymbol{x}) = S_g f(\boldsymbol{x}), \ \forall \boldsymbol{x} \in \mathbb{R}^m, g \in G \tag{1}$$

where $T_g$ and $S_g$ are the transformations. For the SE(3) group, when $m' = 1$, the output of $f$ is a scalar, we have

$$f(T_g\boldsymbol{x}) = f(\boldsymbol{x}), \ \forall \boldsymbol{x} \in \mathbb{R}^m, g \in G \tag{2}$$

thus $f$ is SE(3)-invariant.

**Complete Geometric Representations.**    A geometric transformation $\mathcal{F}(\cdot)$ is complete if for two 3D graphs $G^1 = (\mathcal{V}, \mathcal{E}, \mathcal{P}^1)$ and $G^2 = (\mathcal{V}, \mathcal{E}, \mathcal{P}^2)$, there exists $T_g \in$ SE(3) such that the representations

$$\mathcal{F}(G^1) = \mathcal{F}(G^2) \Longleftrightarrow P_i^1 = T_g(P_i^2), \text{ for } i = 1, \ldots n \tag{3}$$

The operation $T_g$ would not change the 3D conformation of a 3D graph [15, 32, 33]. And $\mathcal{F}(G) \Longleftrightarrow \mathcal{P}$, positions can generate geometric representations, which can also be recovered from them.

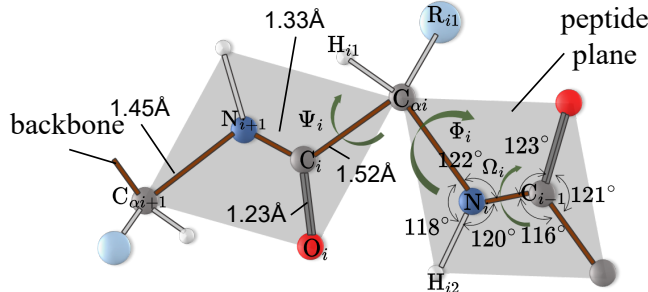

Figure 1: The polypeptide chain depicting the characteristic backbone bond lengths, angles, and torsion angles ($\Psi_i$, $\Phi_i$, $\Omega_i$). The planar peptide groups are denoted as shaded gray regions, indicating that the peptide plane differs from the geometric plane calculated from 3D positions.

**Message Passing Paradigm.** Message passing mechanism is mainly applied in graph convolutional networks (GCNs) [37], which follows an iterative scheme of updating node representations based on the feature aggregation from nearby nodes.

$$
\begin{aligned}
\boldsymbol{h}_i^{(0)} &= \mathrm{BN}\left(\mathrm{FC}\left(\boldsymbol{x}_i\right)\right), \\
\boldsymbol{u}_i^{(l)} &= f_{\mathrm{Agg}}^{(l)}(\boldsymbol{h}_j^{(l-1)}|v_j \in \mathcal{N}(v_i)), \\
\boldsymbol{h}_i^{(l)} &= f_{\mathrm{Update}}^{(l)}(\boldsymbol{h}_j^{(l-1)}, \boldsymbol{u}_i^{(l)})
\end{aligned}
\tag{4}
$$

where $\mathrm{FC}(\cdot)$ and $\mathrm{BN}(\cdot)$ mean the linear transformation and batch normalization respectively. $\mathcal{N}(v_i)$ denotes the neighbours of node $v_i$. $f_{\mathrm{Agg}}^{(l)}$ and $f_{\mathrm{Update}}^{(l)}$ are aggregation and transformation functions at the $l$-th layer, which are permutation invariant and equivariant of node representations.

## 3.2 Sequence-Structure Graph Construction

Specifically, we represent each amino acid as a node, considering the residue types and their positions $i = 1, 2, \cdots, n$ in the sequence, we define the sequential graph primarily on the sequence, if $\|i - j\| < l$, the edge $\varepsilon_{ij}$ exists, where $l$ is a hyper-parameter, and $\|\cdot\|$ denotes the $l^2$-norm. Besides, we predefine a radius $r$, and build the radius graph. There exists a radius edge between node $v_i$ and $v_j$ if $\|d_{ij,\mathrm{C}\alpha}\| < r$, where $d_{ij,\mathrm{C}\alpha} = P_{i,\mathrm{C}\alpha} - P_{j,\mathrm{C}\alpha}$, and $P_{i,\mathrm{C}\alpha}$ denotes the 3D position of $\mathrm{C}_\alpha$ in the $i$-th residue. We dynamically change the predefined thresholds with the depth of the network to cover nodes from the local to the global.

Firstly, we design a base approach at the amino acid level (aa) called $\mathrm{CoupleNet}_{\mathrm{aa}}$ that only uses the $\mathrm{C}_\alpha$ positions of the structures. Inspired by Ingraham et al., we construct a local coordinate system (LCS) for each residue (Figure 7(a) in the appendix).

$$
\boldsymbol{Q}_i = [\boldsymbol{b_i} \quad \boldsymbol{n_i} \quad \boldsymbol{b_i} \times \boldsymbol{n_i}]
\tag{5}
$$

where $\boldsymbol{u}_i = \frac{P_{i,\mathrm{C}\alpha} - P_{i-1,\mathrm{C}\alpha}}{\|P_{i,\mathrm{C}\alpha} - P_{i-1,\mathrm{C}\alpha}\|}$, $\boldsymbol{b_i} = \frac{\boldsymbol{u}_i - \boldsymbol{u}_{i+1}}{\|\boldsymbol{u}_i - \boldsymbol{u}_{i+1}\|}$, $\boldsymbol{n_i} = \frac{\boldsymbol{u}_i \times \boldsymbol{u}_{i+1}}{\|\boldsymbol{u}_i \times \boldsymbol{u}_{i+1}\|}$, $\times$ denotes the vector outer product. Then, we can get the geometric representations at the amino acid level of a protein 3D graph,

$$
\mathcal{F}(G)_{ij,aa} = (\|d_{ij,\mathrm{C}\alpha}\|, \boldsymbol{Q}_i^T \cdot \frac{d_{ij,\mathrm{C}\alpha}}{\|d_{ij,\mathrm{C}\alpha}\|}, \boldsymbol{Q}_i^T \cdot \boldsymbol{Q}_j)
\tag{6}
$$

where $\cdot$ is the matrix multiplication. This implementation is SE(3)-equivariant and obtains complete representations at this level; as if we have $\boldsymbol{Q}_i$, the LCS $\boldsymbol{Q}_j$ can be easily obtained from $\mathcal{F}(G)_{ij,aa}$.

For a node $v_i$, the node features $\boldsymbol{x}_{i,aa}$ at the amino acid level is the concatenation of the one-hot embeddings of amino acid types and the physicochemical properties of each residue, namely, a steric parameter, hydrophobicity, volume, polarizability, isoelectric point, helix probability and sheet probability [39, 40], which provide quantitative insights into the biochemical nature of residues.

Secondly, as illustrated in Figure 1, CoupleNet considers all backbone atoms $\mathrm{C}\alpha, \mathrm{C}, \mathrm{N}, \mathrm{O}$ (as depicted in Figure 2). In detail, the peptide bond displays partial double-bond character due to resonance [41],

indicating that the three non-hydrogen atoms comprising the bond are coplanar, with limited free rotation about the bond due to this coplanar property. The $N_i - C_{\alpha i}$ and $C_{\alpha i} - C_i$ bonds constitute the two bonds in the basic repeating unit of the polypeptide backbone. These single bonds allow unrestricted rotation until sterically restricted by side chains [42, 43]. Since the coordinates of $C_\alpha$ can be obtained as we have the complete representations at the amino acid level, the coordinates of other backbone atoms based on these rigid bond lengths and angles are able to be determined with the remaining degree of the backbone torsion angles $\Phi_i, \Psi_i, \Omega_i$. The omega torsion angle around the $C - N$ peptide bond is typically restricted to nearly $180°$ (trans) but can approach $0°$ (cis) in rare instances. Other than the bond lengths and angles presented in Figure 1, all the H bond lengths measure approximately 1 Å.

For the sequential graph, we compute the sine and cosine values of $\Phi_i, \Psi_i, \Omega_i$ for each amino acid $i$, and also use them as node features for node $v_i$.

$$\boldsymbol{x}_i = \boldsymbol{x}_{i,aa} \| ((\sin \wedge \cos)(\Phi_i, \Psi_i, \Omega_i)) \tag{7}$$

where $\|$ denotes concatenation. There is no isolated node for the designed graph, which means the backbone atoms can be determined one by one along the polypeptide chain based on the positions of $C_\alpha$ and these three backbone dihedral angles. Therefore, the existing presentations $[\mathcal{F}(G)_{ij,aa}]_{i,j=1,...,n}$ and $[\boldsymbol{x}_i]_{i=1,...,n}$ are complete at the backbone level for the sequential graph.

For the radius graph, we want to get the positions of backbone atoms in any two amino acids $i$ and $j$. Inspired by trRosetta [44], the relative rotations and distances are computed, including the distance $(d_{ij,C_\beta})$, three dihedral angles $(\omega_{ij}, \theta_{ij}, \theta_{ji})$ and two planar angles $(\varphi_{ij}, \varphi_{ji})$, as shown in Figure 7(b) in the appendix. These interresidue geometries define the relative locations of the backbone atoms of two residues in their details [44]. Therefore, these six geometries are complete for amino acids at the backbone (bb) level for the radius graph. The graph edges contain the relative spatial information between any two neighboring amino acids $\boldsymbol{e}_{ij} = \mathcal{F}(G)_{ij,aa} \| \mathcal{F}(G)_{ij,bb}$,

$$\mathcal{F}(G)_{ij,bb} = (d_{ij,C_\beta}, (\sin \wedge \cos)(\omega_{ij}, \theta_{ij}, \varphi_{ij})) \tag{8}$$

Protein structures that are SE(3) equivalents have the same 3D conformation, differing in orientation/positioning. Graph representations must encode these structures equivalently since the underlying molecular properties are identical. Constructing the relationships between sequence and structure can help the model learn more comprehensive protein representations, which ensures the model focuses on meaningful aspects of protein structures.

### 3.3 Sequence-Structure Graph Convolution

We employ graph convolution to embed sequences and structures simultaneously, exploring their relationships to generate effective embeddings. Different from previous works [14, 28], we innovatively construct two categories of graphs for sequence and structure and design comprehensive sequential and structural representations to achieve completeness at the amino acid and backbone levels. We then convolve node and edge features aided by the message passing mechanism.

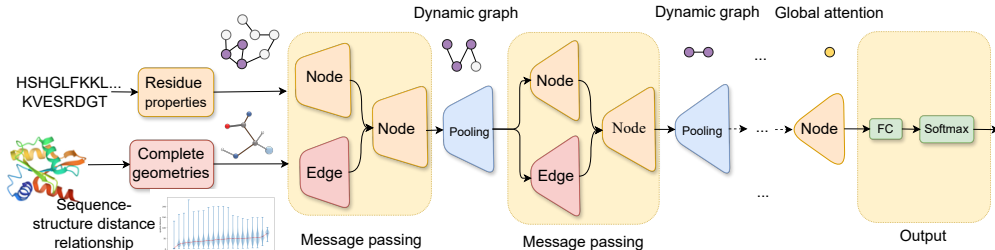

Figure 2: An illustration of CoupleNet. This framework processes protein sequences and structures to get complete geometries and properties used as graph node and edge features, where the sequential and structural graph is dynamically changed depending on their distance relationships and the network depth. Convolutions happen on the nodes and edges simultaneously to capture the relationships from the local to the global.

In order to implement convolution on nodes and edges simultaneously between sequence and structure, we set $\varepsilon_{ij}$ to exist if the following conditions are satisfied:

$$\|i - j\| < l \quad \text{and} \quad \|d_{ij,\text{C}\alpha}\| < r \tag{9}$$

The existing node and edge feature matrices $(X, E)$ are complete representations of a protein 3D graph to reconstruct its backbone atom positions. When the thresholds $r, l$ are small, Eq. 9 defines the local environment [45] of an amino acid, and the structural and chemical properties of a residue are highly dependent on surrounding residues.

Compared with the equation Eq. 4, the proposed CoupleNet first applies a $\text{FC}(\cdot)$ layer and a $\text{BN}(\cdot)$ layer to the node features to obtain the initial encoded representation. Then the aggregation function $f_{\text{Agg}}^{(l)}$ is applied to gather neighboring features of nodes and edges by convolution, where $\sigma(\cdot)$ is the activation function, LeakyReLU. $W$ is the learnable convolutional kernel matrix whose learnable parameters have no concern with the number of nodes or edges. We use the dropout and $\text{FC}(\cdot)$ layer and add a residual connection from the previous layer for update function $f_{\text{Update}}^{(l)}$:

$$
\begin{aligned}
\boldsymbol{h}_i^{(0)} &= \text{BN}\left(\text{FC}\left(\boldsymbol{x}_i\right)\right), \\
\boldsymbol{u}_i^{(l)} &= \sigma(\text{BN}(\sum_{v_j \in \mathcal{N}(v_i)} W \boldsymbol{e}_{ij} \boldsymbol{h}_j^{(l-1)})), \\
\boldsymbol{h}_i^{(l)} &= \boldsymbol{h}_i^{(l)} + \text{Dropout}(\text{FC}(\boldsymbol{u}_i^{(l)}))
\end{aligned}
\tag{10}
$$

By incorporating complete geometric representations to the commonly used message passing framework (Eq. 10), a complete message passing scheme can be achieved [15, 32, 33], which can capture small changes due to such rigid transformations in coordinate positions. Complete representations allow for powerful equivariance and invariance properties to be encoded, which makes the learned models robust. By incorporating complete geometries, the convolution and pooling operations on irregular and non-Euclidean data like graphs are defined and conducted, enabling more expressive modeling for protein data.

### 3.4 Model Architecture

The overall framework of CoupleNet is shown in Figure 2. The inputs to the graph are the calculated sequential and structural representations $(X, E)$. We employ complete message passing and sequence pooling layers to obtain the deeply encoded graph-level representations. After one average pooling layer, the number of residues reduces by half. Thus, we expand the radius $r$ to $2r$ after once pooling, which makes neighbors of center nodes gradually cover more distant and rare nodes, also reducing the computational complexity.

**Differences with Existing Protein Modeling Methods.** The proposed approach representing the sequence and the 3D geometric structure of a protein differs from several existing protein models [12, 14]. Specifically, GearNet [14] has $2l + 1$ types of edges; there are only two different types of graphs in the proposed CoupleNet: radius graph and sequential graph. Importantly, the threshold in the radius graph in GearNet is set to be constant, but we change the threshold of radius dynamically to learn different distance relationships. The message passing mechanism only executes on nodes in CoupleNet instead of on nodes and edges alternately used in GearNet. Moreover, CoupleNet performs convolutions on nodes and edges simultaneously with several pooling layers to reduce the sequence length, which is also largely different from ComENet [33] and GearNet.

**Complexity Analysis.** Considering the computational complexity of one message passing layer in this framework, it is $\mathcal{O}(nd_n)$, where $d_n$ is the average node degree, and $d_n \ll n$. The time complexity is related to the computational complexity of the message passing layer; as we conduct the graph convolution on nodes and edges simultaneously, the time complexity is also $\mathcal{O}(nd_n)$. Assuming there are $m_\varepsilon$ edges in the graph, $d_1$ and $d_2$ mean the feature dimensions of nodes and edges, the space complexity is $\mathcal{O}(nd_1 + m_\varepsilon d_2)$ for every message passing layer. Using $B_s$ to denote the size of the batch, the final computational complexity is only $\mathcal{O}(B_s n d_n)$.

Table 1: Accuracy (%) on fold classification and enzyme reaction classification. [*] denotes the results are taken from [28]. The best and suboptimal results are shown in bold and underline.

| Input | Method | Fold Classification | | | Enzyme |
| | | Fold | SuperFamily | Family | Reaction |
|---|---|---|---|---|---|
| Sequence | ResNet [22]* | 10.1 | 7.21 | 23.5 | 24.1 |
| | Transformer [22]* | 9.22 | 8.81 | 40.4 | 26.6 |
| Structure | 3DCNN_MQA [24]* | 31.6 | 45.4 | 92.5 | 72.2 |
| | IEConv (atom level) [12]* | 45.0 | 69.7 | 98.9 | 87.2 |
| Sequence-Structure | GraphQA [25]* | 23.7 | 32.5 | 84.4 | 60.8 |
| | GVP [26]* | 16.0 | 22.5 | 83.8 | 65.5 |
| | ProNet-Amino Acid [15] | 51.5 | 69.9 | 99.0 | 86.0 |
| | ProNet-Backbone [15] | 52.7 | 70.3 | 99.3 | 86.4 |
| | IEConv (residue level) [12]* | 47.6 | 70.2 | 99.2 | 87.2 |
| | GearNet [14] | 28.4 | 42.6 | 95.3 | 79.4 |
| | GearNet-IEConv [14] | 42.3 | 64.1 | 99.1 | 83.7 |
| | GearNet-Edge [14] | 44.0 | 66.7 | 99.1 | 86.6 |
| | GearNet-Edge-IEConv [14] | 48.3 | 70.3 | 99.5 | 85.3 |
| | CDConv [28] | 56.7 | 77.7 | 99.6 | 88.5 |
| | CoupleNet (Proposed) | **60.6** | **82.1** | **99.7** | **89.0** |

# 4 Experiments

## 4.1 Datasets, Settings and Baselines

Following the tasks in IEconv [12] and Gear-Net [14], we evaluate CoupleNet on four protein tasks: protein fold classification, enzyme reaction classification, GO term prediction, and EC number prediction. For the task of fold and reaction classification, the performance is measured by mean accuracy. For GO Term and EC Prediction, $F_{max}$ is used as the evaluation metric. The performance is measured with mean values of five different initializations. As stated before, we increase the predefined radius $r$ to $2r$ after one pooling layer, from 4 to 16, and $l$ is set to be a constant number 11, and the number of feature channels is also doubled. In this condition, when the number of nodes decreases, $l$ is constant, $r$ increases and neighbors of center nodes gradually cover distant nodes. We design the sequential and radius graph instead of the $k$-nearest neighbor graph because a constant

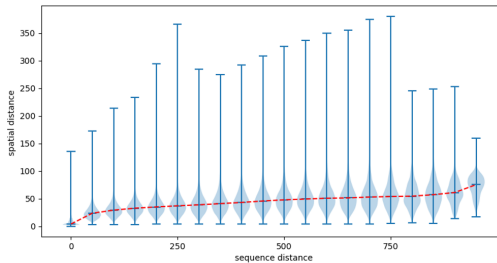

Figure 3: The violin plot of the relationships of distances between sequence and structure on the GO term prediction dataset, the sequential distance $\|i - j\|$ is from 1 to $n$-1, and the x-axis means sequential distance subtract one, the y-axis means $d_{ij,C\alpha}$. The dashed red line connects the median values.

$k$ makes some neighboring nodes far away from the center node. Distances of a group of neighbor nodes are larger than 20 Å, which cannot be seen as contacts [46].

We present the dataset statistics Table 4 and conduct experiments to analyze these datasets. Figure 3 shows the distance relationships between sequence and structure on the GO term training dataset with 29898 proteins. We can see that when the sequential distance is large, there still exist nodes spatially adjacent. According to the trend of the medians, when the sequence is long, atoms may need more space to arrange in the 3D space. The violin plot distance relationships on the other three datasets are presented in Figure 8.

We compare our proposed method with existing protein representation learning methods, which are classified into three categories based on their inputs: a sequence (amino acid sequence), 3D structure, or both sequence and structure. 1) Sequence-based encoders, including CNN [47],

Table 2: $F_{max}$ on GO term and EC number prediction. [*] means the results are taken from [28]. The best and suboptimal results are shown in bold and underline.

| Category | Method | GO-BP | GO-MF | GO-CC | EC |
|---|---|---|---|---|---|
| Sequence | ResNet [22]* | 0.280 | 0.405 | 0.304 | 0.605 |
| | Transformer [22]* | 0.264 | 0.211 | 0.405 | 0.238 |
| Structure | GCN [37]* | 0.252 | 0.195 | 0.329 | 0.320 |
| | GAT [48]* | 0.284 | 0.317 | 0.385 | 0.368 |
| | 3DCNN_MQA [24]* | 0.240 | 0.147 | 0.305 | 0.077 |
| Sequence-Structure | GraphQA [25]* | 0.308 | 0.329 | 0.413 | 0.509 |
| | GVP [26]* | 0.326 | 0.426 | 0.420 | 0.489 |
| | IEConv (residue level) [12]* | 0.421 | 0.624 | 0.431 | - |
| | GearNet [14] | 0.356 | 0.503 | 0.414 | 0.730 |
| | GearNet-IEConv [14] | 0.381 | 0.563 | 0.422 | 0.800 |
| | GearNet-Edge [14] | 0.403 | 0.580 | 0.450 | 0.810 |
| | GearNet-Edge-IEConv [14] | 0.400 | 0.581 | 0.430 | 0.810 |
| | CDConv [28] | 0.453 | 0.654 | 0.479 | 0.820 |
| | CoupleNet (Proposed) | **0.467** | **0.669** | **0.494** | **0.866** |

ResNet [22], LSTM [22] and Transformer [22]. 2) Structure-based methods (GCN [37], GAT [48], 3DCNN_MQA [24], IEConv (atom level) [12]). 3) Sequence-structure based models, e.g., GVP [26], ProNet [15], GearNet [14], CDConv [28], etc.

## 4.2 Results of Fold and Reaction Classification, EC and GO term Prediction.

Table 1 provides the comparisons on the fold and enzyme reaction classification. From this table, we can see that the proposed model CoupleNet achieves the best performance across all four test sets on the fold and enzyme reaction classification compared with recent state-of-the-art methods. Especially on the Fold and SuperFamily test sets, CoupleNet improves the results by about $4\%$, showing that CoupleNet is proficient at learning the mappings between protein sequences, structures, and functions. Moreover, CDConv [28] ranks second among these methods. Both CDConv and our method are implemented by sequence-structure convolution. This phenomenon illustrates that coupling sequences and structures of proteins are conducive to learning better protein embeddings. Our proposed model utilizes complete geometric representations and specially designed dynamically changed graphs, achieving state-of-the-art results.

We follow the split method in [14, 17] to guarantee that the test set only comprises PDB chains with sequence identity no higher than $95\%$ to the training set for GO term and EC number prediction. Proteins are organized into three ontologies: molecular function (MF), biological process (BP), and cellular component (CC) for the task of GO term prediction. Table 2 compares different protein modeling methods on GO term prediction and EC number prediction. The proposed model, CoupleNet yields the highest $F_{max}$ across these four test sets of two tasks, outperforming other prevalent models. This indicates that CoupleNet can effectively predict the functions, locations, and enzymatic activities of proteins.

We learn protein representations in terms of protein sequences and structures, which is essential as building hierarchical dependencies gets universal representations when there is a low similarity between the training and test sets. We compare the protein graph methods, GearNet, and the proposed CoupleNet by different cutoff splits. Proteins in the test set are categorized into four groups based on their similarity to the training set (30%, 40%, 50%, 70%), not by the default split rate (95%). The results are shown in Figure 9 in the appendix, which indicate that even when there is a low similarity between the training and test sets, our model also has the highest scores, which demonstrates the superiority and robustness of the proposed model.

## 4.3 Analysis of Experiments on Protein Length

Table 3: Ablation of CoupleNet, we compare it with the base model, $\text{CoupleNet}_{\text{aa}}$, and the models removing the sequence (w/o sequence), structure, or related geometries.

| Method | Fold Classification | | | Enzyme | GO | | | EC |
|---|---|---|---|---|---|---|---|---|
| | Fold | SuperFamily | Family | Reaction | BP | MF | CC | |
| CoupleNet | 60.6 | 82.1 | 99.7 | 89.0 | 0.467 | 0.669 | 0.494 | 0.866 |
| $\text{CoupleNet}_{\text{aa}}$ | 57.8 | 78.7 | 99.6 | 88.6 | 0.458 | 0.660 | 0.484 | 0.851 |
| w/o sequence | 60.0 | 81.6 | 99.6 | 88.4 | 0.441 | 0.650 | 0.456 | 0.700 |
| w/o structure | 26.1 | 36.4 | 92.9 | 81.3 | 0.406 | 0.586 | 0.427 | 0.625 |
| w/o $\Phi, \Psi, \Omega$ | 60.3 | 81.3 | 99.6 | 88.7 | 0.463 | 0.666 | 0.490 | 0.862 |
| w/o $d, \omega, \theta, \varphi$ | 60.4 | 81.5 | 99.7 | 88.9 | 0.461 | 0.666 | 0.488 | 0.864 |

To assess the model's proficiency in handling proteins of varying lengths, we conducted a categorization based on the sequence length of proteins using the mean length of the unseen test set as a threshold. This allowed us to distinguish between small and large proteins. For instance, proteins were grouped into two categories based on the mean length: 149.4 for Fold, 186.7 for SuperFamily, and 162.4 for Family, 299.8 for the GO dataset.

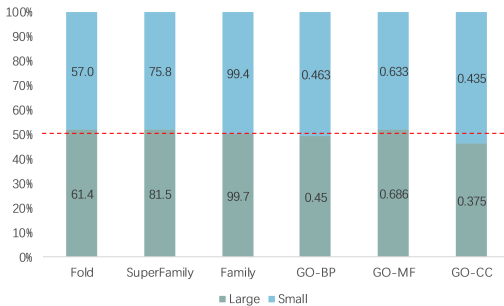

Figure 4: Percentage accumulation chart of results on large and small proteins. The vertical axis shows the percentage, where the red line indicates 50%.

The protein Fold classification task involves identifying remote homology relationships (i.e., determining if proteins belong to the same family or superfamily), CoupleNet demonstrates its capability to capture such relationships. Notably, higher accuracies are observed for relatively large proteins with sequence lengths surpassing the mean length, as indicated in Figure 4. On the other hand, the GO term prediction task exhibits less correlation with protein sequence length, which relies more on the local environment of residues [49].

## 4.4 Ablation Study

We examined the impact of removing the sequence information, which means removing the encoding of amino acid types for each node; removing the structure information, which means removing features related to protein geometry ($\mathcal{F}(G)_{aa}, \Phi, \Psi, \Omega, d_{C_\beta}, \omega, \theta, \varphi$, subscripts are omitted for brevity); removing the backbone torsion angles (w/o $\Phi, \Psi, \Omega$) and removing the interresidue geometric structure representations (w/o $d_{C_\beta}, \omega, \theta, \varphi$). As shown in Table 3, removing either sequence or structure causes a performance drop on all tasks, demonstrating that both types of information are critical for the proposed method. When removing the structure, the performance decreases more significantly, suggesting that structural information provides more important and comprehensive clues compared with sequence information alone. From these tables, we can also find that these complete geometries ($\Phi, \Psi, \Omega$ and $d_{C_\beta}, \omega, \theta, \varphi$) provide similar information, with one of their removals leading to minor performance drops for the reason that they both provide complete geometries, but from different perspectives. Compared with $\text{CoupleNet}_{\text{aa}}$, CoupleNet achieves significant improvements on the four tasks, demonstrating the importance of complete representations at the backbone level in learning protein embeddings.

## 5 Conclusion and Limitation

In this work, we propose CoupleNet, a novel protein representation learning method that dynamically fuses protein sequences and multi-level structures by conducting convolution on graph nodes and edges. We design the sequential and radius graphs, achieving completeness on them at different protein structure levels. Our approach achieves state-of-the-art results on the protein tasks, demonstrating the superiority of our proposed method. A limitation is that this framework needs protein sequences and structures to be available, which may not be suitable for the sequence-only data as the input. A future direction is to develop a large-scale model based on the multi-modal protein data types to enhance performance.

## Acknowledgements

This work was supported by the National Science and Technology Major Project (No. 2022ZD0115101), the National Natural Science Foundation of China Project (No. U21A20427 and No. 623B2086), Project (No. WU2022A009) from the Center of Synthetic Biology and Integrated Bioengineering of Westlake University and Project (No. WU2023C019) from the Westlake University Industries of the Future Research Funding.

## Footnotes

*Equal contribution. †Correspondence: {stan.zq.li}@westlake.edu.cn

## References

[1] Andrew W Senior, Richard Evans, John Jumper, James Kirkpatrick, Laurent Sifre, Tim Green, Chongli Qin, Augustin Žídek, Alexander WR Nelson, Alex Bridgland, et al. Improved protein structure prediction using potentials from deep learning. *Nature*, 577(7792):706–710, 2020.

[2] Bin Ma. Novor: real-time peptide de novo sequencing software. *Journal of the American Society for Mass Spectrometry*, 26(11):1885–1894, 2015.

[3] UniProt Consortium. Uniprot: a worldwide hub of protein knowledge. *Nucleic acids research*, 47(D1):D506–D515, 2019.

[4] Noelia Ferruz and Birte Höcker. Controllable protein design with language models. *Nature Machine Intelligence*, 4(6):521–532, 2022.

[5] Bozhen Hu, Jun Xia, Jiangbin Zheng, Cheng Tan, Yufei Huang, Yongjie Xu, and Stan Z Li. Protein language models and structure prediction: Connection and progression. *arXiv preprint arXiv:2211.16742*, 2022.

[6] Yue Liu, Xiaoxin He, Miao Xiong, Jinlan Fu, Shumin Deng, and Bryan Hooi. Flipattack: Jailbreak llms via flipping. *arXiv preprint arXiv:2410.02832*, 2024.

[7] Ahmed Elnaggar, Michael Heinzinger, Christian Dallago, Ghalia Rehawi, Yu Wang, Llion Jones, Tom Gibbs, Tamas Feher, Christoph Angerer, Martin Steinegger, et al. Prottrans: Toward understanding the language of life through self-supervised learning. *IEEE transactions on pattern analysis and machine intelligence*, 44(10):7112–7127, 2021.

[8] Alexander Rives, Joshua Meier, Tom Sercu, Siddharth Goyal, Zeming Lin, Jason Liu, Demi Guo, Myle Ott, C Lawrence Zitnick, Jerry Ma, et al. Biological structure and function emerge from scaling unsupervised learning to 250 million protein sequences. *Proceedings of the National Academy of Sciences*, 118(15):e2016239118, 2021.

[9] Zeming Lin, Halil Akin, Roshan Rao, Brian Hie, Zhongkai Zhu, Wenting Lu, Allan dos Santos Costa, Maryam Fazel-Zarandi, Tom Sercu, Sal Candido, et al. Language models of protein sequences at the scale of evolution enable accurate structure prediction. *BioRxiv*, 2022: 500902, 2022.

[10] John Jumper, Richard Evans, Alexander Pritzel, Tim Green, Michael Figurnov, Olaf Ronneberger, Kathryn Tunyasuvunakool, Russ Bates, Augustin Žídek, Anna Potapenko, et al. Highly accurate protein structure prediction with alphafold. *Nature*, 596(7873):583–589, 2021.

[11] Mihaly Varadi, Stephen Anyango, Mandar Deshpande, Sreenath Nair, Cindy Natassia, Galabina Yordanova, David Yuan, Oana Stroe, Gemma Wood, Agata Laydon, et al. Alphafold protein structure database: massively expanding the structural coverage of protein-sequence space with high-accuracy models. *Nucleic acids research*, 50(D1):D439–D444, 2022.

[12] Pedro Hermosilla, Marco Schäfer, Matěj Lang, Gloria Fackelmann, Pere Pau Vázquez, Barbora Kozlíková, Michael Krone, Tobias Ritschel, and Timo Ropinski. Intrinsic-extrinsic convolution and pooling for learning on 3d protein structures. *arXiv preprint arXiv:2007.06252*, 2020.

[13] Pedro Hermosilla and Timo Ropinski. Contrastive representation learning for 3d protein structures. *arXiv preprint arXiv:2205.15675*, 2022.

[14] Zuobai Zhang, Minghao Xu, Arian Jamasb, Vijil Chenthamarakshan, Aurelie Lozano, Payel Das, and Jian Tang. Protein representation learning by geometric structure pretraining. In *The Eleventh International Conference on Learning Representations*, 2023.

[15] Limei Wang, Haoran Liu, Yi Liu, Jerry Kurtin, and Shuiwang Ji. Learning hierarchical protein representations via complete 3d graph networks. In *The Eleventh International Conference on Learning Representations*, 2023.

[16] Fang Wu, Dragomir Radev, and Jinbo Xu. When geometric deep learning meets pretrained protein language models. *bioRxiv*, pages 2023–01, 2023.

[17] Vladimir Gligorijević, P Douglas Renfrew, Tomasz Kosciolek, Julia Koehler Leman, Daniel Berenberg, Tommi Vatanen, Chris Chandler, Bryn C Taylor, Ian M Fisk, Hera Vlamakis, et al. Structure-based protein function prediction using graph convolutional networks. *Nature communications*, 12(1):3168, 2021.

[18] Cheng Tan, Zhangyang Gao, Jun Xia, Bozhen Hu, and Stan Z Li. Generative de novo protein design with global context. *arXiv preprint arXiv:2204.10673*, 2022.

[19] Meng Liu, Zhengyang Wang, and Shuiwang Ji. Non-local graph neural networks. *IEEE transactions on pattern analysis and machine intelligence*, 44(12):10270–10276, 2021.

[20] Kaixiong Zhou, Xiao Huang, Yuening Li, Daochen Zha, Rui Chen, and Xia Hu. Towards deeper graph neural networks with differentiable group normalization. *Advances in neural information processing systems*, 33:4917–4928, 2020.

[21] Dušan Petrović, Valeria A Risso, Shina Caroline Lynn Kamerlin, and Jose M Sanchez-Ruiz. Conformational dynamics and enzyme evolution. *Journal of the Royal Society Interface*, 15 (144):20180330, 2018.

[22] Roshan Rao, Nicholas Bhattacharya, Neil Thomas, Yan Duan, Peter Chen, John Canny, Pieter Abbeel, and Yun Song. Evaluating protein transfer learning with tape. *Advances in neural information processing systems*, 32, 2019.

[23] Roshan M Rao, Jason Liu, Robert Verkuil, Joshua Meier, John Canny, Pieter Abbeel, Tom Sercu, and Alexander Rives. Msa transformer. In *International Conference on Machine Learning*, pages 8844–8856. PMLR, 2021.

[24] Georgy Derevyanko, Sergei Grudinin, Yoshua Bengio, and Guillaume Lamoureux. Deep convolutional networks for quality assessment of protein folds. *Bioinformatics*, 34(23):4046–4053, 2018.

[25] Federico Baldassarre, David Menéndez Hurtado, Arne Elofsson, and Hossein Azizpour. Graphqa: protein model quality assessment using graph convolutional networks. *Bioinformatics*, 37(3):360–366, 2021.

[26] Bowen Jing, Stephan Eismann, Patricia Suriana, Raphael JL Townshend, and Ron Dror. Learning from protein structure with geometric vector perceptrons. *arXiv preprint arXiv:2009.01411*, 2020.

[27] Sheng Chen, Zhe Sun, Lihua Lin, Zifeng Liu, Xun Liu, Yutian Chong, Yutong Lu, Huiying Zhao, and Yuedong Yang. To improve protein sequence profile prediction through image captioning on pairwise residue distance map. *Journal of chemical information and modeling*, 60(1):391–399, 2019.

[28] Hehe Fan, Zhangyang Wang, Yi Yang, and Mohan Kankanhalli. Continuous-discrete convolution for geometry-sequence modeling in proteins. In *The Eleventh International Conference on Learning Representations*, 2023.

[29] Ruijie Quan, Wenguan Wang, Fan Ma, Hehe Fan, and Yi Yang. Clustering for protein representation learning. *arXiv preprint arXiv:2404.00254*, 2024.

[30] Can Chen, Jingbo Zhou, Fan Wang, Xue Liu, and Dejing Dou. Structure-aware protein self-supervised learning. *Bioinformatics*, 39(4):btad189, 2023.

[31] Zeyuan Wang, Qiang Zhang, Hu Shuangwei, Haoran Yu, Xurui Jin, Zhichen Gong, and Huajun Chen. Multi-level protein structure pre-training via prompt learning. In *The Eleventh International Conference on Learning Representations*, 2023.

[32] Yi Liu, Limei Wang, Meng Liu, Yuchao Lin, Xuan Zhang, Bora Oztekin, and Shuiwang Ji. Spherical message passing for 3d molecular graphs. In *International Conference on Learning Representations*, 2021.

[33] Limei Wang, Yi Liu, Yuchao Lin, Haoran Liu, and Shuiwang Ji. Comenet: Towards complete and efficient message passing for 3d molecular graphs. *arXiv preprint arXiv:2206.08515*, 2022.

[34] Justin Gilmer, Samuel S Schoenholz, Patrick F Riley, Oriol Vinyals, and George E Dahl. Neural message passing for quantum chemistry. In *International conference on machine learning*, pages 1263–1272. PMLR, 2017.

[35] Fabian Fuchs, Daniel Worrall, Volker Fischer, and Max Welling. Se (3)-transformers: 3d roto-translation equivariant attention networks. *Advances in Neural Information Processing Systems*, 33:1970–1981, 2020.

[36] Weitao Du, He Zhang, Yuanqi Du, Qi Meng, Wei Chen, Nanning Zheng, Bin Shao, and Tie-Yan Liu. Se (3) equivariant graph neural networks with complete local frames. In *International Conference on Machine Learning*, pages 5583–5608. PMLR, 2022.

[37] Thomas N Kipf and Max Welling. Semi-supervised classification with graph convolutional networks. *arXiv preprint arXiv:1609.02907*, 2016.

[38] John Ingraham, Vikas Garg, Regina Barzilay, and Tommi Jaakkola. Generative models for graph-based protein design. *Advances in neural information processing systems*, 32, 2019.

[39] Gang Xu, Qinghua Wang, and Jianpeng Ma. Opus-rota4: a gradient-based protein side-chain modeling framework assisted by deep learning-based predictors. *Briefings in Bioinformatics*, 23(1):bbab529, 2022.

[40] Jack Hanson, Kuldip Paliwal, Thomas Litfin, Yuedong Yang, and Yaoqi Zhou. Improving prediction of protein secondary structure, backbone angles, solvent accessibility and contact numbers by using predicted contact maps and an ensemble of recurrent and residual convolutional neural networks. *Bioinformatics*, 35(14):2403–2410, 2019.

[41] Manfred S Weiss, Andreas Jabs, and Rolf Hilgenfeld. Peptide bonds revisited. *Nature structural biology*, 5(8):676–676, 1998.

[42] David L Nelson, Albert L Lehninger, and Michael M Cox. *Lehninger principles of biochemistry*. Macmillan, 2008.

[43] K Peter C Vollhardt and Neil E Schore. *Organic chemistry: structure and function*. Macmillan, 2003.

[44] Jianyi Yang, Ivan Anishchenko, Hahnbeom Park, Zhenling Peng, Sergey Ovchinnikov, and David Baker. Improved protein structure prediction using predicted interresidue orientations. *Proceedings of the National Academy of Sciences*, 117(3):1496–1503, 2020.

[45] Bin Huang, Tingwen Fan, Kaiyue Wang, Haicang Zhang, Chungong Yu, Shuyu Nie, Yangshuo Qi, Wei-Mou Zheng, Jian Han, Zheng Fan, Shiwei Sun, Sheng Ye, Huaiyi Yang, and Dongbo Bu. Accurate and efficient protein sequence design through learning concise local environment of residues. *Bioinformatics*, 39(3):btad122, 03 2023.

[46] G Marius Clore and Angela M Gronenborn. Nmr structure determination of proteins and protein complexes larger than 20 kda. *Current opinion in chemical biology*, 2(5):564–570, 1998.

[47] Amir Shanehsazzadeh, David Belanger, and David Dohan. Is transfer learning necessary for protein landscape prediction? *arXiv preprint arXiv:2011.03443*, 2020.

[48] Petar Veličković, Guillem Cucurull, Arantxa Casanova, Adriana Romero, Pietro Lio, and Yoshua Bengio. Graph attention networks. *arXiv preprint arXiv:1710.10903*, 2017.

[49] Ilya Levental and Ed Lyman. Regulation of membrane protein structure and function by their lipid nano-environment. *Nature Reviews Molecular Cell Biology*, 24(2):107–122, 2023.

[50] Jie Hou, Badri Adhikari, and Jianlin Cheng. Deepsf: deep convolutional neural network for mapping protein sequences to folds. *Bioinformatics*, 34(8):1295–1303, 2018.

[51] Edwin C Webb et al. *Enzyme nomenclature 1992. Recommendations of the Nomenclature Committee of the International Union of Biochemistry and Molecular Biology on the Nomenclature and Classification of Enzymes.* Number Ed. 6. Academic Press, 1992.

[52] Marina V Omelchenko, Michael Y Galperin, Yuri I Wolf, and Eugene V Koonin. Non-homologous isofunctional enzymes: a systematic analysis of alternative solutions in enzyme evolution. *Biology direct*, 5:1–20, 2010.

[53] Helen M Berman, John Westbrook, Zukang Feng, Gary Gilliland, Talapady N Bhat, Helge Weissig, Ilya N Shindyalov, and Philip E Bourne. The protein data bank. *Nucleic acids research*, 28(1):235–242, 2000.

[54] Diederik P Kingma and Jimmy Ba. Adam: A method for stochastic optimization. *arXiv preprint arXiv:1412.6980*, 2014.

# A    Protein Sequence and Structure

There exist relationships between sequence and structure, as shown in Figure 5; the contact maps can be inferred from sequences to obtain 3D coordinates, while the distant contacts provide constraints for a multiple sequence alignment (MSA). Besides, protein sequence and structure provide different types of information (Figure 6), like residue types and 3D arrangements.

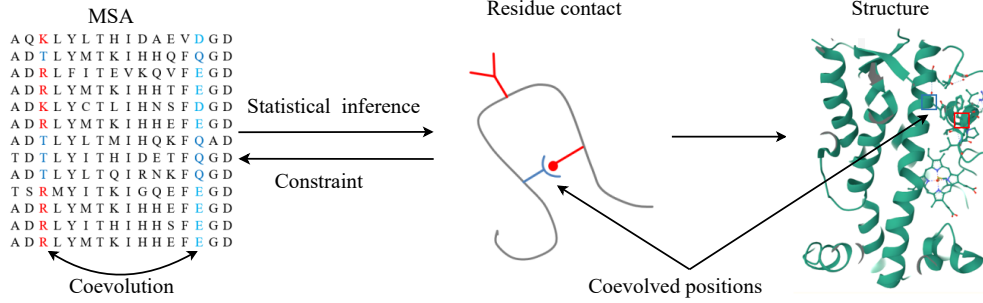

Figure 5: Relationship between protein sequences and the structure of one protein in the alignment. The positions that coevolved are highlighted in red and light blue. Residues within these positions where changes occurred are shown in blue. Given such a MSA, one can infer correlations statistically found between two residues that these sequence positions are spatially adjacent, i.e., they are contacts [5]. The protein tertiary structure can be inferred from such contacts.

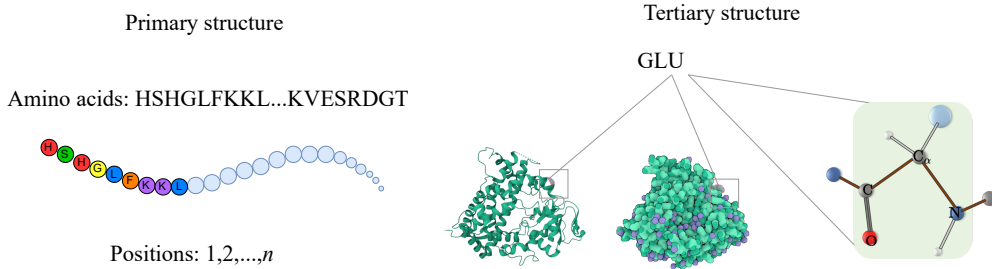

Figure 6: Illustration of protein sequence and structure. 1) The primary structure comprises $n$ amino acids. 2) The tertiary structure with atom arrangement in Euclidean space is presented, where each atom has a specific 3D coordinate. Amino acids have fixed backbone atoms ($C_\alpha, C, N, O$) and side-chain atoms that vary depending on the residue types.

# B    SE(3)

The collection of $4 \times 4$ real matrices of the SE(3) is shown as:

$$\begin{bmatrix} R & \mathbf{t} \\ 0 & 1 \end{bmatrix} = \begin{bmatrix} r_{11} & r_{12} & r_{13} & t_1 \\ r_{21} & r_{22} & r_{23} & t_2 \\ r_{31} & r_{32} & r_{33} & t_3 \\ 0 & 0 & 0 & 1 \end{bmatrix}, \tag{11}$$

where $R \in \mathrm{SO}(3)$ and $\mathbf{t} \in \mathbb{R}^3$, SO(3) is the 3D rotation group. $R$ satisfying $R^T R = I$ and $\det(R) = 1$.

# C    Geometric Features

Firstly, we present the local coordinate system, which is shown in Figure 7(a), $b_i$ is the negative bisector of the angle between the rays ($P_{i-1,C_\alpha} - P_{i,C_\alpha}$) and ($P_{i+1,C_\alpha} - P_{i,C_\alpha}$).

Secondly, we introduce the relative rotations and distances in trRosetta [44], including the distance ($d_{ij,C_\beta}$), three dihedral angles ($\omega_{ij}, \theta_{ij}, \theta_{ji}$) and two planar angles ($\varphi_{ij}, \varphi_{ji}$), as shown in Figure 7(b).

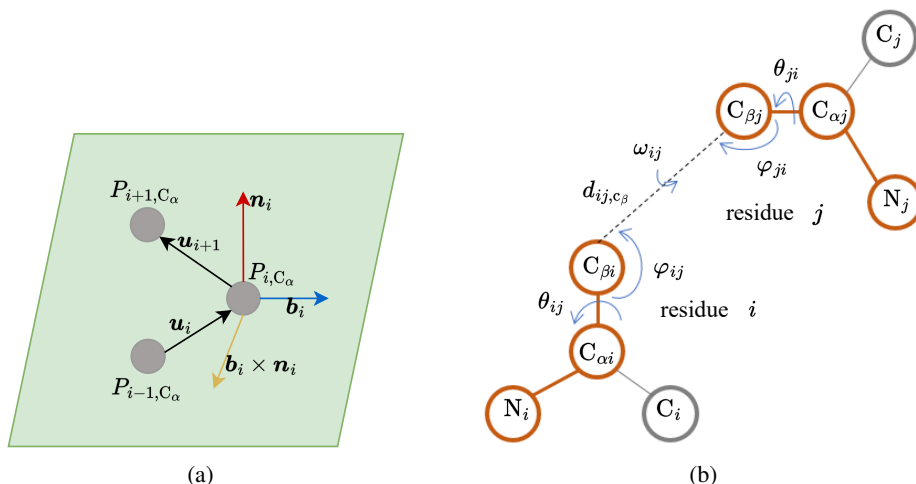

Figure 7: Protein geometries. (a) The local coordinate system. $P_{i,C_\alpha}$ is the coordinate of $C_\alpha$ in residue $i$. (b) Interresidue geometries including angles and distances, including the distance ($d_{ij,C_\beta}$), three dihedral angles ($\omega_{ij}, \theta_{ij}, \theta_{ji}$) and two planar angles ($\varphi_{ij}, \varphi_{ji}$).

# D    Social Impact

The development of CoupleNet marks a significant advancement in the realm of protein research with profound potential for societal impact. By effectively bridging the gap between protein sequences and structures, CoupleNet offers unprecedented insights into the intricate world of proteins and their biological functions. This breakthrough has far-reaching implications for various fields, including medicine and biotechnology. With the ability to accurately predict protein functions, CoupleNet can expedite drug discovery processes, enabling the development of novel therapeutics for combating diseases more efficiently. Furthermore, its capability to identify infrequently encountered functions and capture remote homology relationships promises to revolutionize our understanding of protein evolution and diversity, paving the way for innovative solutions in fields such as personalized medicine and bioprocessing. Ultimately, the deployment of CoupleNet has the potential to drive transformative advancements in healthcare and biotechnology, ultimately leading to improved human health, enhanced crop yields, and sustainable industrial practices.

# E    Tasks

**Fold Classification.**    Protein fold is to predict the fold class label given a protein, which is crucial for understanding how protein structure and protein evolution interact [50]. In total, this dataset contains $16,712$ proteins with $1,195$ fold classes. There are three test sets available. Fold: Training excludes proteins from the same superfamily. Superfamily: Training does not include proteins from the same family. Family: Proteins from the same family are included in the training.

**Enzyme Reaction Classification.**    Reaction categorization aims to predict a protein's class of enzyme-catalyzed reactions, according to all four levels of the EC number [51, 52]. Following the setting in [12], this dataset has $37,248$ proteins from 384 four-level EC numbers [53].

**GO Term Prediction.**    The goal of GO term prediction is to foretell whether a protein is related to a certain GO term. Following [17], The three ontologies, MF, BP, and CC, are hierarchically connected, functional classes. MF describes activities that occur at the molecular level, BP represents the larger processes, and CC describes the parts of a cell or its extracellular environment [3].

**EC Number Prediction.**    This task seeks to predict the 538 EC numbers from the third level and fourth levels of different proteins [17], which describe their catalysis of biochemical reactions.

# F  Details of Datasets and Training Setup

For all datasets, we use a data augmentation strategy by adding noise to the training set to increase the variability of data. For example, we update the position of $C_{\alpha i}$,

$$P_{i,C_\alpha} \leftarrow P_{i,C_\alpha} + N(\mu_N, \sigma_N^2) \tag{12}$$

where $\mu_N, \sigma_N^2$ are the mean (expectation) and variance of the normal distribution, which are set to 0 and 0.1 in experiments. Dataset statistics of our four downstream tasks are summarized in Table 4.

Table 4: Dataset statistics. #X means the number of X.

| Dataset | #Train | #Validation | #Test |
|---|---|---|---|
| EC | $15,550$ | $1,729$ | $1,919$ |
| GO | $29,898$ | $3,322$ | $3,415$ |
| Fold | $12,312$ | $736$ | $718$ |
| Superfamily | $12,312$ | $736$ | $1,254$ |
| Family | $12,312$ | $736$ | $1,272$ |
| Reaction Classification | $29,215$ | $2,562$ | $5,651$ |

**Settings.**  The proposed models are trained with the Adam optimizer [54], which are conducted on a single NVIDIA-SMI A100 GPU through PyTorch 1.13+cu117 and PyTorch Geometric 2.3.1 with CUDA 11.2. The number of the initial feature channels is 256. The learning rate is set to 0.001. The radius threshold increases from 4 to 16, and $l$ is set to 11. For the number of pooling layers that are related to the depth of the network, we employ the grid search trick to search suitable hyperparameters for them. There are four pooling layers that are sufficient to achieve satisfactory results; every two message-passing layers are followed by an average pooling layer. Thus, there are eight message-passing layers. In detail, the sequence average pooling functions perform customized average pooling operations on the input tensors based on the calculated indices (dividing the length of the sequence by 2 and flooring the result). It aggregates and summarizes information from the input tensors using scatter operations to produce the output tensors. As for the batch size and training epochs, etc., which influence the convergence speed of deep learning models, we also adopt the grid search method to get a group of values; the details are shown in Table 5.

Table 5: More details of training setup

| Hyper-parameter | Fold | Enzyme Reaction | GO | EC |
|---|---|---|---|---|
| Batch size | 4 | 4 | 24 | 64 |
| Epoch | 400 | 400 | 500 | 500 |

# G  Evaluation Metric

$F_{\max}$ is calculated by first determining the precision and recall for each protein, then averaging these results over all proteins [14, 17, 28]. $p_i^j$ is the prediction probability for the $j$-th class of the $i$-th protein, given the decision threshold $t \in [0, 1]$, the precision and call are give as:

$$\text{precision}_i(t) = \frac{\sum_j \mathbb{I}[((p_i^j \geq t) \cap b_i^j)]}{\sum_j \mathbb{I}[(p_i^j \geq t)]}$$

$$\text{recall}_i(t) = \frac{\sum_j \mathbb{I}[((p_i^j \geq t) \cap b_i^j)]}{\sum_j b_i^j} \tag{13}$$

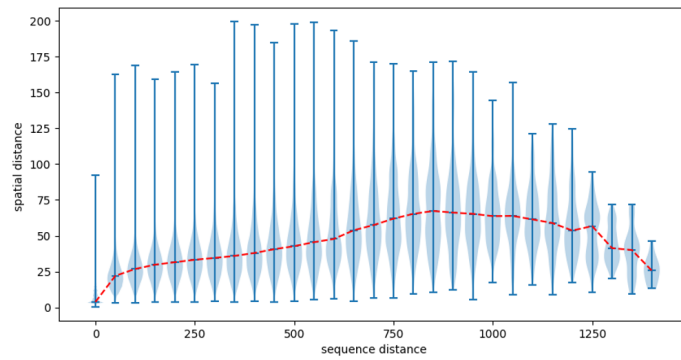

(a) Fold

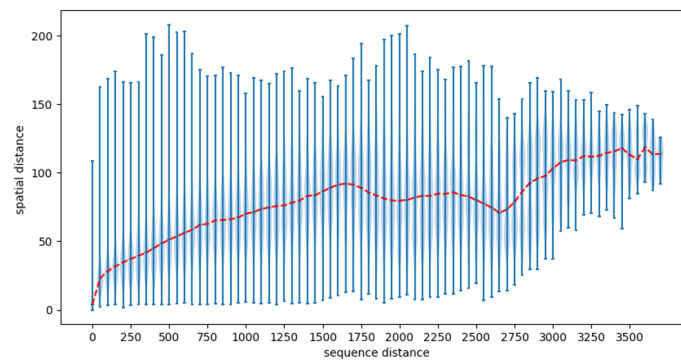

(b) Enzyme Reaction

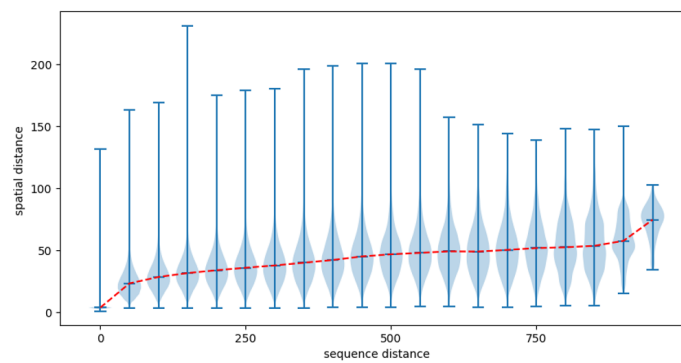

(c) EC

Figure 8: The violin plot of the relationships of distances between sequence and structure on the fold and enzyme reaction classification and EC number prediction datasets.

where $b_i^j \in \{0, 1\}$ is the corresponding binary class label, and $\mathbb{I} \in \{0, 1\}$ is an indicator function. If there are $N$ proteins in total, then the average precision and recall are defined as:

$$\text{precision}(t) = \frac{\sum_i^N \text{precision}_i(t)}{\sum_i^N \left( \left( \sum_j \left( p_i^j \geq t \right) \right) \geq 1 \right)}$$
$$\text{recall}(t) = \frac{\sum_i^N \text{recall}_i(t)}{N}$$
(14)

Finally, $F_{\max}$ is defined as the maximum value of F-score over all thresholds,

$$F_{\max} = \max_t \left\{ \frac{2 \cdot \text{precision}(t) \cdot \text{recall}(t)}{\text{precision}(t) + \text{recall}(t)} \right\}$$
(15)

## H  Completeness Analysis

Given a protein 3D graph $G = (\mathcal{V}, \mathcal{E}, \mathcal{P})$, we capture the geometric representations based on the atoms' 3D positions and use sequential and structural representations as the node and edge features. For a 3D structure, based on the definition of completeness and the rigorously demonstrated method to show the calculated geometries can achieve completeness for structures [15], we guarantee the completeness of the selected geometric representations at the amino acid and backbone levels of structures.

The geometric representations are SE(3) invariant (distances, angles) and SE(3) equivariant (directions, orientations). Therefore, it is natural for Eq. 3 to hold from right to left. To demonstrate Eq. 3 holding from left to right, we need to show $\mathcal{F}(G) \Rightarrow T_g(\mathcal{P})$, where $T_g$ does not change the 3D conformation of a 3D graph. Thus, we need to show positions can be determined by $\mathcal{F}(G)$.

The base approach $\text{CoupleNet}_{\text{aa}}$ only considers the $C_\alpha$ coordinates and constructs LCS for each residue. $\mathcal{F}(G)_{aa}$ provides complete representations. First, when $n = 1$, it holds. Assume the case $n = k$ holds such that $\mathcal{F}(G)_{aa}$ is complete. Then, we need to prove the case $n = k + 1$ still holds. This is obvious because if $v_j$ is the $(k + 1)$-th node connected to node $v_i$ among the existing $k$ nodes, the LCS $Q_j$ can be easily obtained from $Q_i$ and $\mathcal{F}(G)_{aa}$.

When considering the backbone atoms $C_\alpha, C, N, O$, $\mathcal{F}(G)_{aa}$ is complete. As shown in Figure 1, the remaining degree of freedom at the backbone level is the rotation angles $\Phi_i, \Psi_i, \Omega_i$ based on the rigid bond lengths and angles. Such backbone torsion angles are calculated and concatenated with $x_{i,aa}$ into $x_i$. Besides, for any residues $i$ and $j$, the calculated six inter-residue geometries fully define the relative locations of backbone atoms. Therefore, there are no other remaining degrees of freedom. Consequently, the obtained geometric representations at the backbone level are complete.

## I  Analysis of Imbalanced Class Experiments

The classes in the protein fold classification task tend to be imbalanced. This dataset contains 16,712 proteins with 1,195 fold classes, including three categories: Fold, SuperFamily, and Family. We provide the top 10 most frequent classes and the mean prediction accuracies on Fold, SuperFamily, and Family test sets. The results are presented in Table 6, where #Class means the total number of samples of this class. From the table, we can find that the proposed model, CoupleNet, is not heavily affected by the category unbalance problem. For example, there are 1,254 proteins with 1,187 classes in the SuperFamily dataset. A large number of classes have only one protein sample, but we got a mean accuracy of 82.1 percent on SuperFamily.

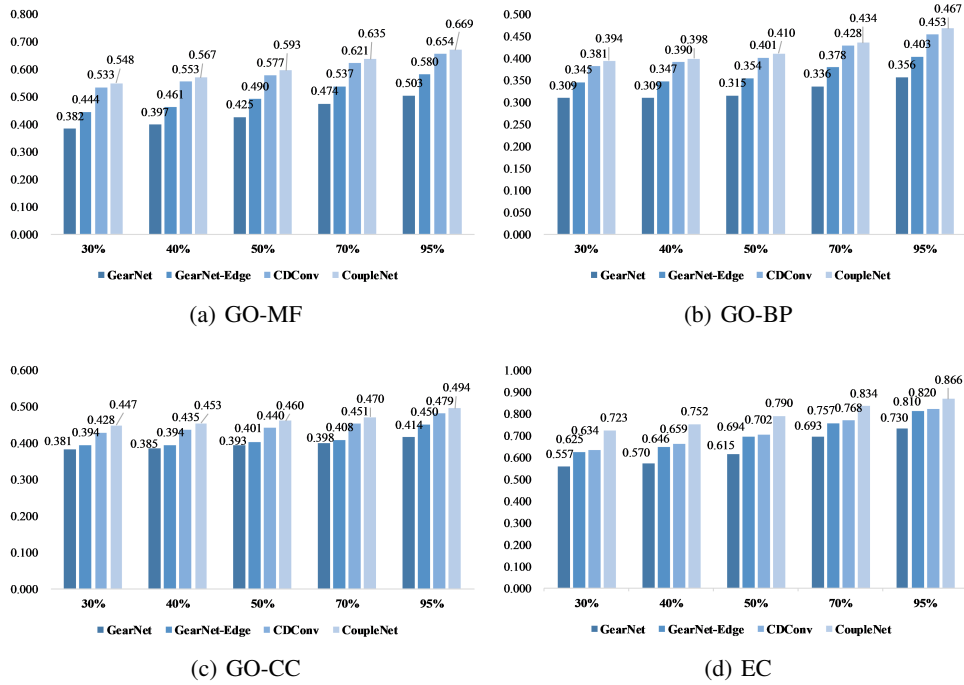

Figure 9: $F_{max}$ on GO term and EC number prediction under different cutoffs.

Table 6: Accuracy (%) on fold classification, the top 10 most frequent classes, and the mean prediction accuracies are listed.

| Fold | | | SuperFamily | | | Family | | |
|---|---|---|---|---|---|---|---|---|
| Class | #Class | Accuracy | Class | #Class | Accuracy | Class | #Class | Accuracy |
| 661 | 30 | 86.7 | 3 | 36 | 1.0 | 284 | 181 | 1.0 |
| 482 | 28 | 64.3 | 661 | 34 | 94.1 | 491 | 34 | 1.0 |
| 458 | 28 | 96.4 | 458 | 29 | 93.1 | 458 | 29 | 1.0 |
| 1068 | 20 | 85.0 | 25 | 23 | 78.3 | 459 | 23 | 1.0 |
| 284 | 20 | 80.0 | 284 | 20 | 1.0 | 1107 | 20 | 1.0 |
| 364 | 20 | 15.0 | 328 | 19 | 1.0 | 661 | 19 | 1.0 |
| 1 | 16 | 81.3 | 546 | 19 | 1.0 | 619 | 19 | 1.0 |
| 400 | 15 | 73.3 | 519 | 18 | 1.0 | 3 | 18 | 1.0 |
| 1107 | 14 | 78.6 | 509 | 16 | 1.0 | 501 | 16 | 1.0 |
| 23 | 14 | 71.4 | 547 | 16 | 1.0 | 321 | 16 | 1.0 |

